# NEURAL ANALOG DIFFUSION-ENHANCEMENT LAYER AND SPATIO-TEMPORAL GROUPING IN EARLY VISION

Allen M. Waxman[*,†], Michael Seibert[*,†], Robert Cunningham[†] and Jian Wu[*]

[*] Laboratory for Sensory Robotics
Boston University
Boston, MA 02215

[†] Machine Intelligence Group
MIT Lincoln Laboratory
Lexington, MA 02173

## ABSTRACT

A new class of neural network aimed at early visual processing is described; we call it a Neural Analog Diffusion-Enhancement Layer or "NADEL." The network consists of two levels which are coupled through feedfoward and shunted feedback connections. The lower level is a two-dimensional diffusion map which accepts visual features as input, and spreads activity over larger scales as a function of time. The upper layer is periodically fed the activity from the diffusion layer and locates local maxima in it (an extreme form of contrast enhancement) using a network of local comparators. These local maxima are fed back to the diffusion layer using an on-center/off-surround shunting anatomy. The maxima are also available as output of the network. The network dynamics serves to cluster features on multiple scales as a function of time, and can be used in a variety of early visual processing tasks such as: extraction of corners and high curvature points along edge contours, line end detection, gap filling in contours, generation of fixation points, perceptual grouping on multiple scales, correspondence and path impletion in long-range apparent motion, and building 2-D shape representations that are invariant to location, orientation, scale, and small deformation on the visual field.

## INTRODUCTION

Computer vision is often divided into two main stages, "early vision" and "late vision", which correspond to image processing and knowledge-based recognition/interpretation, respectively. Image processing for early vision involves algorithms for feature enhancement and extraction (e.g. edges and corners), feature grouping (i.e., perceptual

We acknowledge support from the Machine Intelligence Group of MIT Lincoln Laboratory. The views expressed are those of the authors and do not reflect the official policy or position of the U.S. Government.

organization), and the extraction of physical properties for object surfaces that comprise a scene (e.g. reflectance, depth, surface slopes and curvatures, discontinuities). The computer vision literature is characterized by a plethora of algorithms to achieve many of these computations, though they are hardly robust in performance.

Biological neural network processing does, of course, achieve all of these early vision tasks, as evidenced by psychological studies of the preattentive phase of human visual processing. Often, such studies provide motivation for new algorithms in computer vision. In contrast to this algorithmic approach, computational neural network processing tries to glean organizational and functional insights from the biological realizations, in order to emulate their information processing capabilities. This is desirable, mainly because of the adaptive and real-time nature of the neural network architecture. Here, we shall demonstrate that a single neural architecture based on *dynamical diffusion-enhancement networks* can realize a large variety of early vision tasks that deal mainly with perceptual grouping. The ability to group image features on multiple scales as a function of time, follows from "attractive forces" that emerge from the network dynamics. We have already implemented the NADEL (in 16-bit arithmetic) on a video-rate parallel computer, the PIPE [Kent *et al.*, 1985], as well as a SUN-3 workstation.

## THE NADEL

The *Neural Analog Diffusion-Enhancement Layer* was recently introduced by Seibert & Waxman [1989], and is illustrated in Figure 1; it consists primarily of two levels which are coupled via feedforward and shunted feedback connections. Low-level features extracted from the imagery provide input to the lower level (a 2-D map) which spreads input activity over larger scales as time progresses via diffusion, allowing for passive decay of activity. The diffused activity is periodically sampled and passed upward to a contrast-enhancing level (another 2-D map) which locates local maxima in the terrain of diffuse activity. However, this forward pathway is masked by receptive fields which pass only regions of activity with positive Gaussian curvature and negative mean curvature; that is these receptive fields play the role of inhibitory dendro-dendritic modulatory gates. This masking fascilitates the local maxima detection in the upper level. These local maxima detected by the upper level are fed back to the lower diffusion level using a shunting dynamics with on-center/off-surround anatomy (cf. [Grossberg, 1973] on the importance of shunting for automatic gain control, and the role of center/surround anatomies in competitive networks). The local maxima are also available as outputs of the network, and take on different interpretations as a function of the input. A number of examples of spatio-temporal grouping will be illustrated in the next section.

The primary result of diffusion-enhancement network dynamics is to *create a long-range attractive force between isolated featural inputs*. This force manifests itself by shifting the local maxima of activity toward one another, leading to a featural grouping over multiple scales as a function of time. This is shown in Figure 1, where two featural inputs spread their initial excitations over time. The individual activities superpose, with the tail of one gaussian crossing the maximum of the other gaussian at an angle. This

biases the superposition of activities, adding more activity to one side of a maximum than another, causing a shift in the local maxima toward one another. Eventually, the local maxima merge into a single maximum at the centroid of the individual inputs. If we keep track of the local maxima as diffusion progresses (by connecting the output of the enhancement layer to another layer which stores activity in short term memory), then the two initial inputs will become connected by a line. In Figure 1 we also illustrate the grouping of five features in two clusters, a configuration possessing two spatial scales. After little diffusion the local maxima are located where the initial inputs were. Further diffusion causes each cluster to form a single local maximum at the cluster centroid. Eventually, both clusters merge into a single hump of activity with one maximum at the centroid of the five initial inputs. Thus, multiscale grouping over time emerges. The examples of Figure 1 use only diffusion without any feedback, yet they illustrate the importance of localizing the local maxima through a kind of contrast-enhancement on another layer. The local maxima of activity serve as "place tokens" representing grouped features at a particular scale. The feedback pathway re-activates the diffusion layer, thereby allowing the grouping process to proceed to still larger scales, even across featureless areas of imagery.

The dynamical evolution of activity in the NADEL can be modeled using a modified diffusion equation [Seibert & Waxman, 1989]. However, in our simulations of the NADEL we don't actually solve this differential equation directly. Instead, each iteration of the NADEL consists of a spreading of activity using gaussian convolution, allowing for passive decay, then sampling the diffusion layer, masking out areas which are not positive Gaussian curvature and negative mean curvature activity surfaces, detecting one local maximum in each of these convex areas, and feeding this back to the diffusion layer with a shunted on-center/off-surround excitation at the local maxima. In the biological system, diffusion can be accomplished via a recurrent network of cells with off-center/on-surround lateral connectivity, or more directly using electrotonic coupling across gap junctions as in the horizontal cell layer of the retina [Dowling, 1987]. Curvature masking of the activity surface can be accomplished using oriented off-center/on-surround receptive fields that modulate the connections between the two primary layers of the NADEL.

## SPATIO-TEMPORAL GROUPING

We give several examples of grouping phenomena in early vision, utilizing the NADEL. In all cases its parameters correspond to gaussian spreading with $\sigma=3$ and passive decay of 1% per iteration, and on-center/off-surround feedback with $\sigma_+=1/\sqrt{2}$ and $\sigma_-=1$.

*Grouping of Two Points:* The simple case of two instantaneous point stimuli input simultaneously to the NADEL is summarized in Figure 2. We plot the time ($N$ network iterations) it takes to merge the two inputs, as a function of their initial separation ($S$ pixels). For $S\leq6$ the points merge in one iteration; for $S>24$ activity equilibrates and shifting of local maxima never begins.

*Grouping on Multiple Scales:* Figure 3 illustrates the hierarchy of groupings generated by a square outline (31 pixels on a side) with gaps (9 pixels wide). Corner and line-end features are first enhanced using complementary center-surround receptive fields (modeled as a rectified response to a difference-of-gaussians), and located at the local maxima of activity. These features are shown superimposed on the shape in 3a; they serve as input to the NADEL. Figure 3b shows the loci of local maxima determined up to the second stable grouping, superimposed over the shape. Boundary completion fills the gaps in the square. In Figure 3c we show the loci of local maxima on the image plane, after the final grouping has occured ($N=100$ iterations). The trajectory of local maxima through space-time (x,y,t) is shown in Figure 3d after the fourth grouping. It reveals a hierarchical organization similar to the "scale-space diagrams" of Witkin [1983].

It can be seen from Figure 3d that successive groupings form *stable* entities in that the place tokens remain stationary for several iterations of the NADEL. It isn't until activity has diffused farther out to the next representative scale that these local maxima start moving once again, and eventually merge. *This relates stable perceptual groupings to place tokens (i.e., local maxima of activity) that are not in motion on the diffusion layer.* The motion of place tokens can be measured in the same fashion as feature point motion across the visual field. Real-time receptive fields for measuring the motion of image edge and point features have recently been developed by Waxman *et al.* [1988].

*Grouping of Time-Varying Inputs:* The simplest example in this case corresponds to the grouping of two lights that are flashed at different locations at different times. When the time interval between flashes (Stimulus Onset Asynchrony *SOA*) is set appropriately, one perceives a smooth motion or "path impletion" between the stimuli. This percept of "long-range apparent motion" is the cornerstone of the Gestalt Psychology moverment, and has remained unexplained for one-hundred years now [Kolers, 1972]. We have applied the NADEL to a variety of classical problems in apparent motion including the "split motion" percept and the multi-point Ternus configuration [Waxman *et al.*, 1989].

Here we consider only the case of motion between two stimuli, where we interpret the locus of local maxima as the impleted path in apparent motion. However, the *direction* of perceived motion is not determined by the grouping process itself; only the path. We make the additional assumption that grouping generates a motion percept only if the second stimulus begins to shift immediately upon input to the NADEL. We suggest that *the motion percept occurs only after path impletion is complete.* That is, while grouping is active, its outputs are suppressed from our perception (a form of "transient-on-sustained inhibition" analogous to saccadic suppression). By varying the separation between the two stimuli, and the time (SOA) between their inputs, we can plot regimes for which the NADEL predicts apparent motion. This is shown in Figure 4, which compares favorably with the psychophysical results summarized in Figure 3.2 of [Kolers, 1972]. We find regimes in which complete paths are formed ("smooth motion"), partial paths are formed ("jumpy motion"), and no immediate shifting occurs ("no motion"). The maximum allowable SOA between stimuli (upper curves) is determined by the passive decay rate. Increasing this decay from 1% to 3% will decrease the maximum SOA by a

factor of five. The minimum allowable SOA (lower curves) increases with increasing separation, since it takes longer for activity from the first stimulus to influence a more distant second stimulus. The linearity of the lower boundary has been interpreted by [Waxman *et al.*, 1989] as suggestive of Korte's "third law" [Kolers, 1972], when taken in combination with a logarithmic transformation of the visual field [Schwartz, 1980].

*Attentional Cues and Invariant Representations:* Place tokens which emerge as stable groupings over time can also provide attentional cues to a vision system. They would typically drive saccadic eye motions during scene inspection, with the relative activities of these maxima and their order of emergence determining the sequence of rapid eye motions. Such eye motions are known to play a key role in human visual perception [Yarbus, 1967]; they are influenced by both bottom-up perceptual cues as well as top-down expectations. The neuromorphic vision system developed by Seibert & Waxman [1989], shown in Figure 5, utilizes the NADEL to drive "eye motions", and thereby achieve translational invariance in 2-D object learning and recognition. This is followed by a log-polar transform (which emulates the geniculo-cortical connections [Schwartz, 1980]) and another NADEL to achieve rotation and scale invariance as well. Further coding of the transformed feature points by overlapping receptive fields provides invariance to small deformation. Pattern learning and recognition is then achieved using an Adaptive Resonance Theory (ART-2) network [Carpenter & Grossberg, 1987].

# REFERENCES

G. Carpenter & S. Grossberg (1987). ART-2: Self-organization of stable category recognition codes for analog input patterns. *Applied Optics* 26, pp. 4919-4930.

J.E. Dowling (1987). **The RETINA: An approachable part of the brain.** Cambridge, MA: Harvard University Press.

S. Grossberg (1973). Contour enhancement, short term memory, and constancies in reverberating neural networks. *Studies in Applied Mathematics* 52, pp. 217-257.

E.W. Kent, M.O. Shneier & R. Lumia (1985). PIPE: Pipelined Image Processing Engine. *Journal of Parallel and Distributed Computing* 2, pp. 50-78.

P.A. Kolers (1972). **Aspects of Motion Perception.** New York: Pergamon Press.

E.L. Schwartz (1980). Computational anatomy and functional architecture of striate cortex: A spatial mapping approach to perceptual coding. *Vision Research* 20, pp. 645-669.

M. Seibert & A.M. Waxman (1989). Spreading activation layers, visual saccades and invariant representations for neural pattern recognition systems. *Neural Networks* 2, pp. 9-27.

A.M. Waxman, J. Wu & F. Bergholm (1988). Convected activation profiles and the measurement of visual motion. *Proceeds. 1988 IEEE Conference on Computer Vision and Pattern Recognition*, Ann Arbor, MI, pp. 717-723.

A.M. Waxman, J. Wu & M. Seibert (1989). Computing visual motion in the short and the long: From receptive fields to neural networks. *Proceeds. IEEE 1989 Workshop on Visual Motion*, Irvine, CA.

A.P. Witkin (1983). Scale space filtering. *Proceeds. of the International Joint Conference on Artificial Intelligence*, Karlsruhe, pp. 1019-1021.

A.L. Yarbus (1967). **Eye Movements and Vision.** New York: Plenum Press.

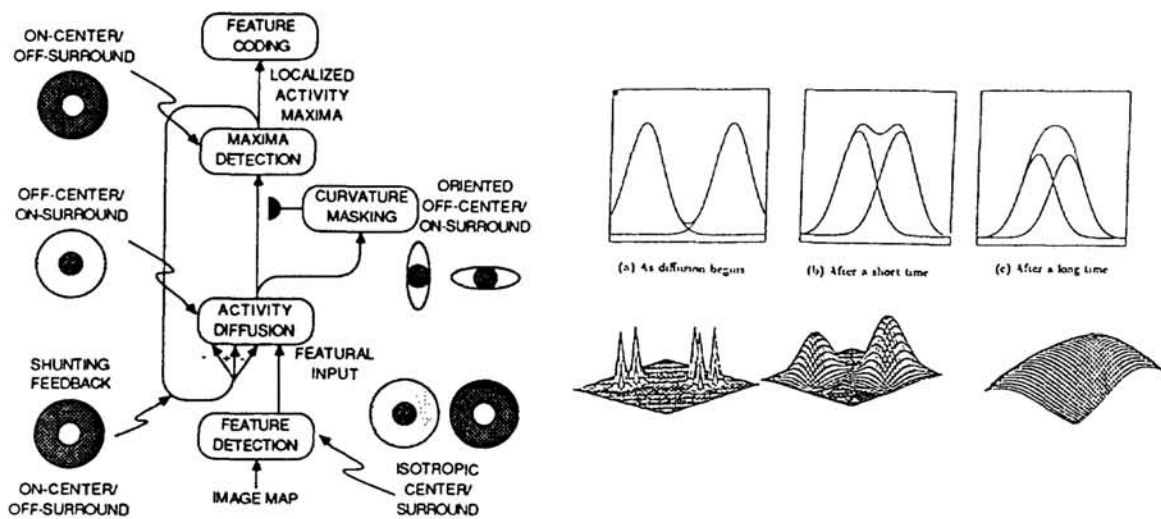

Figure 1 - (left) The NADEL takes featural input and diffuses it over a 2-D map as a function of time. Local maxima of activity are detected by the upper layer and fed back to the diffusion layer using an on-center/off-surround shunting anatomy. (right, top) Features spread their activities which superpose to generate an attractive force, causing distant features to group. (right, bottom) Grouping progresses in time over multiple scales, with small clusters emerging before extended clusters. Clusters are represented by their local maxima of activity, which serve as place tokens.

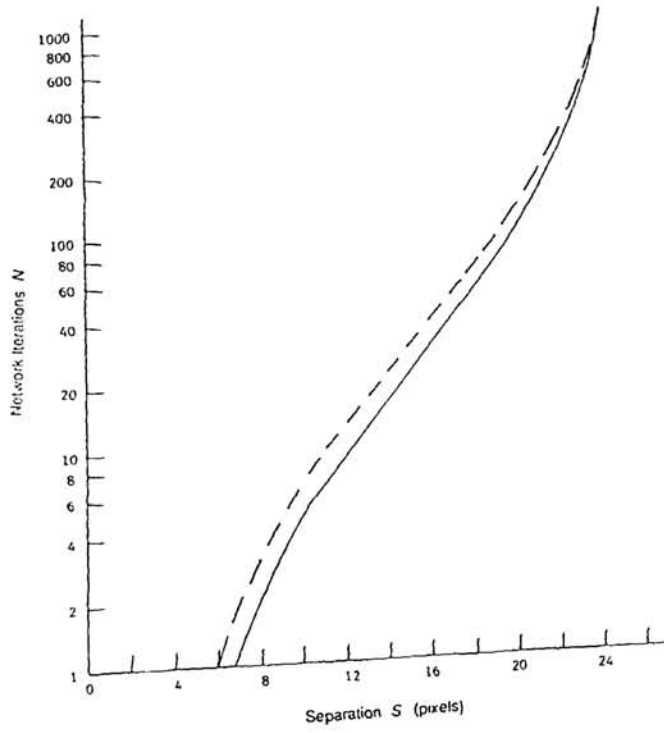

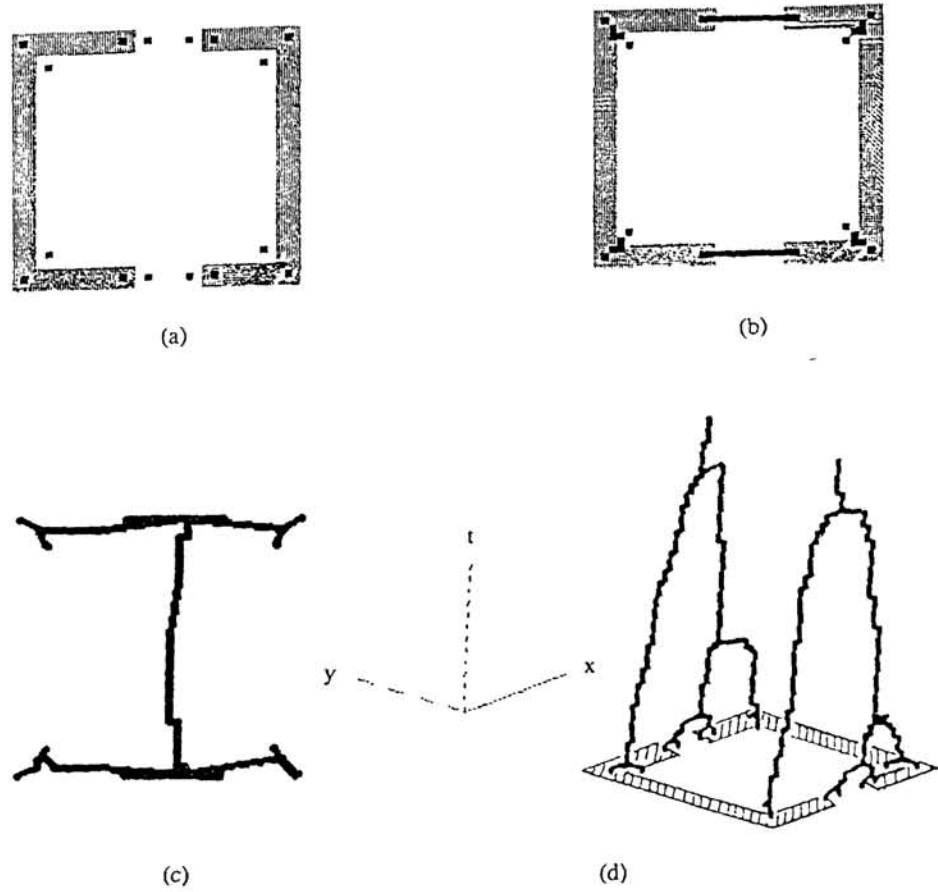

(a)

(b)

(c)

(d)

Figure 2 - The time (network iterations $N$) to merge two points input simultaneously to the NADEL, as a function of their initial separation ($S$ pixels).

Figure 3 - Perceptual grouping of a square outline with gaps.

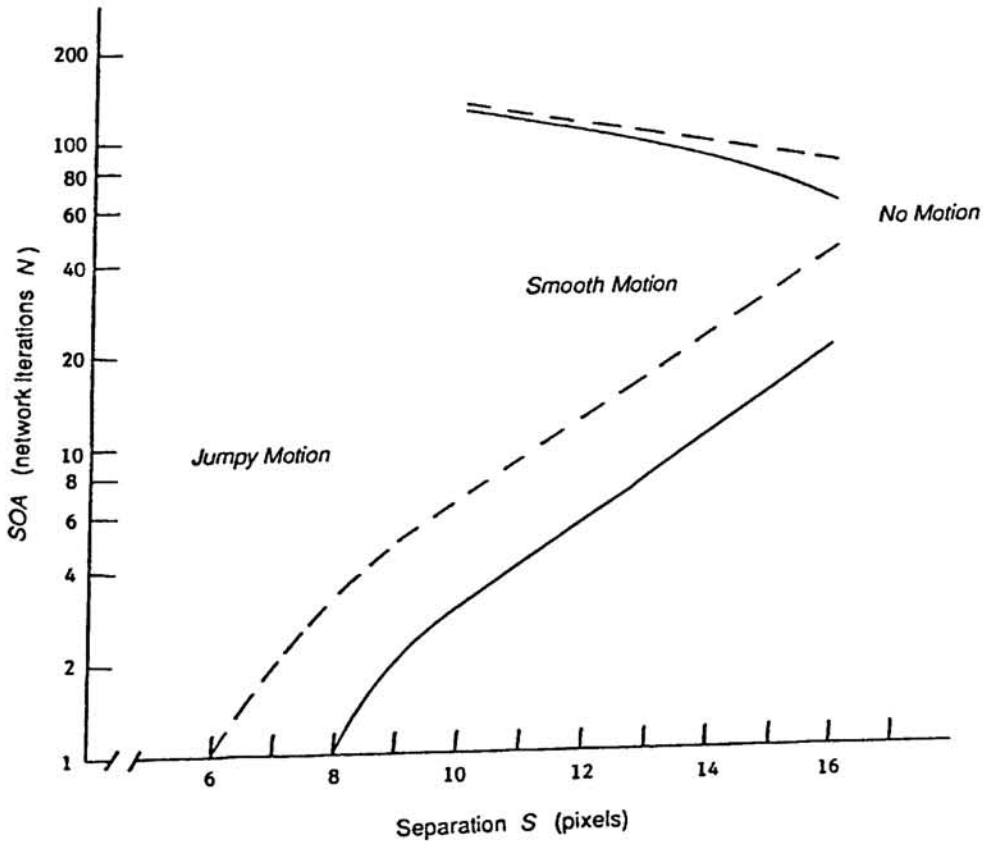

Figure 4 - Apparent motion between two flashed lights: Stimulus Onset Asynchrony *SOA* (network iterations *N*) vs. Separation *S* (pixels). Solid curves indicate boundaries between which introduction of the second light yields immediate shifting of local maxima; dashed curves (above solid curves) indicate when final merge occurs yielding the impleted path.

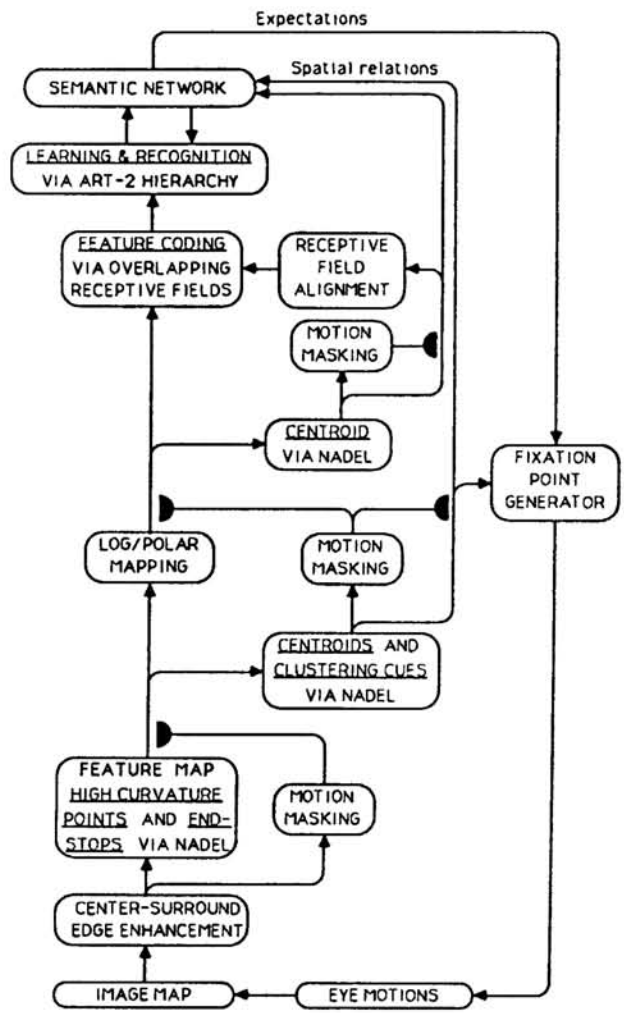

Figure 5 - The neuromorphic vision system for invariant learning and recognition of 2-D objects utilizes three NADEL networks.